# Learning Human-like Knowledge by Singular Value Decomposition: A Progress Report

**Thomas K. Landauer**          **Darrell Laham**
Department of Psychology & Institute of Cognitive Science
University of Colorado at Boulder    Boulder, CO 80309-0345
{landauer, dlaham}@psych.colorado.edu

**Peter Foltz**
Department of Psychology
New Mexico State University    Las Cruces, NM 88003-8001
pfoltz@crl.nmsu.edu

## Abstract

Singular value decomposition (SVD) can be viewed as a method for unsupervised training of a network that associates two classes of events reciprocally by linear connections through a single hidden layer. SVD was used to learn and represent relations among very large numbers of words (20k-60k) and very large numbers of natural text passages (1k-70k) in which they occurred. The result was 100-350 dimensional "semantic spaces" in which any trained or newly added word or passage could be represented as a vector, and similarities were measured by the cosine of the contained angle between vectors. Good accuracy in simulating human judgments and behaviors has been demonstrated by performance on multiple-choice vocabulary and domain knowledge tests, emulation of expert essay evaluations, and in several other ways. Examples are also given of how the kind of knowledge extracted by this method can be applied.

## 1  INTRODUCTION

Traditionally, imbuing machines with human-like knowledge has relied primarily on explicit coding of symbolic facts into computer data structures and algorithms. A serious limitation of this approach is people's inability to access and express the vast reaches of unconscious knowledge on which they rely, knowledge based on masses of implicit inference and irreversibly melded data. A more important deficiency of this state of affairs is that by coding the knowledge ourselves, (as we also do when we assign subjectively hypothesized rather than objectively identified features to input or output nodes in a neural net) we beg important questions of how humans acquire and represent the coded knowledge in the first place.

Thus, from both engineering and scientific perspectives, there are reasons to try to design learning machines that can acquire human-like quantities of human-like knowledge from the same sources as humans. The success of such techniques would not prove that the same mechanisms are used by humans, but because we presently do not know how the problem can be solved in principle, successful simulation may offer theoretical insights as well as practical applications. In the work reported here we have found a way to induce significant amounts of knowledge about the meanings of passages and of their constituent vocabularies of words by training on large bodies of natural text. In general terms, the method simultaneously extracts the similarity between words (the likelihood of being used in passages that convey similar ideas) and the similarity of passages (the likelihood of containing words of similar meaning). The conjoint estimation of similarity is accomplished by a fundamentally simple representational technique that exploits mutual constraints implicit in the occurrences of very many words in very many contexts. We view the resultant system both as a means for automatically learning much of the semantic content of words and passages, and as a potential computational model for the process that underlies the corresponding human ability.

While the method starts with data about the natural contextual co-occurrences of words, it uses them in a different manner than has previously been applied. A long-standing objection to co-occurrence statistics as a source of linguistic knowledge (Chomsky's 1957) is that many grammatically acceptable expressions, for example sentences with potentially unlimited embedding structures, cannot be produced by a finite Markov process whose elements are transition probabilities from word to word. If word-word probabilities are insufficient to generate language, then, it is argued, acquiring estimates of such probabilities cannot be a way that language can be learned. However, our approach to statistical knowledge learning differs from those considered in the past in two ways. First, the basic associational data from which knowledge is induced are not transition frequencies between successive individual words or phrases, but rather the frequencies with which particular words appear as components of relatively large natural passages, utterances of the kind that humans use to convey complete ideas. The result of this move is that the statistical regularities reflected are relations among unitary expressions of meaning, rather than syntactic constraints on word order that may serve additional purposes such as output and input processing efficiencies, error protection, or esthetic style. Second, the mutual constraints inherent in a multitude of such local co-occurrence relations are jointly satisfied by being forced into a global representation of lower dimensionality. This constraint satisfaction, a form of induction, was accomplished by singular value decomposition, a linear factorization technique that produces a representational structure equivalent to a three layer neural network model with linear activation functions.

## 2  THE TEXT ANALYSIS MODEL AND METHOD

The text analysis process that we have explored is called Latent Semantic Analysis (LSA) (Deerwester et al., 1990; Landauer and Dumais, 1997). It comprises four steps:

(1) A large body of text is represented as a matrix [ij], in which rows stand for individual word types, columns for meaning-bearing passages such as sentences or paragraphs, and cells contain the frequency with which a word occurs in a passage.

(2) Cell entries ($freq_{ij}$) are transformed to:

$$\frac{\log(freq_{ij}+1)}{-\sum_{1-j}\left(\left(\frac{freq_{ij}}{\sum_{1-j}freq_{ij}}\right)*\log\left(\frac{freq_{ij}}{\sum_{1-j}freq_{ij}}\right)\right)}$$

a measure of the first order association of a word and its context.

(3) The matrix is then subjected to singular value decomposition (Berry, 1992):

$$[ij] = [ik] \ [kk] \ [jk]'$$

in which [ik] and [jk] have orthonormal columns, [kk] is a diagonal matrix of singular values, and $k <= \max (i,j)$.

(4) Finally, all but the *d* largest singular values are set to zero. Pre-multiplication of the right-hand matrices produces a least-squares best approximation to the original matrix given the number of dimensions, *d*, (hidden units in a corresponding neural net model representation) that are retained. The SVD with dimension reduction constitutes a constraint-satisfaction induction process in that it predicts the original observations on the basis of linear relations among the abstracted representations of the data that it has retained. By hypothesis, the analysis induces human-like relationships among passages and words because humans also make inferences about semantic relationships from abstracted representations based on limited data, and do so by an analogous process.

In the result, each word and passage is represented as a vector of length *d*. Performance depends strongly on the choice of number of dimensions. The optimal number is typically around 300. The similarity of any two words, any two text passages, or any word and any text passage, are computed by measures on their vectors. We have most often used the cosine (of the contained angle between the vectors in semantic *d*-space) — which we interpret as the degree of qualitative similarity of meaning. The length of vectors is also useful and interpretable.

# 3 TESTS OF LSA'S PERFORMANCE

LSA's ability to simulate human knowledge and meaning relations has been tested in a variety of ways. Here we describe two relatively direct sources of evidence and briefly list several others.

## 3.1 VOCABULARY & DOMAIN KNOWLEDGE TESTS

In all cases, LSA was first trained on a large text corpus intended to be representative of the text from which humans gain most of the semantic knowledge to be simulated. In a previously reported test (Landauer and Dumais, 1997), LSA was trained on approximately five million words of text sampled from a high-school level encyclopedia, then tested on multiple choice items from the Educational Testing Service Test of English as a Foreign Language (TOEFL). These test questions present a target word or short phrase and ask the student to choose the one of four alternative words or phrases that is most similar in meaning. LSA's answer was determined by computing the cosine between the derived vector for the target word or phrase and each of the alternatives and choosing the largest. LSA was correct on 64% of the 80 items, identical to the average of a large sample of students from non-English speaking countries who had applied for admission to U. S. colleges. When in error, LSA made choices positively correlated (product-moment $r = .44$) with those preferred by students. We have recently replicated this result with training on a similar sized sample from the Associated Press newswire

In a new set of tests, LSA was trained on a popular introductory psychology textbook (Myers, 1995) and tested with the same four-alternative multiple choice tests used for students in two large classes. In these experiments, LSA's score was about 60%—lower than the class averages but above passing level, and far above guessing probability. Its errors again resembled those of students; it got right about half as many of questions rated difficult by the test constructors as ones rated easy (Landauer, Foltz and Laham, 1997).

## 3.2 ESSAY TESTS

Word-word meaning similarities are a good test of knowledge—indeed, vocabulary tests are the best single measure of human intelligence. However, they are not sufficient to assess the correspondence of LSA and human knowledge because people usually express knowledge via larger verbal strings, such as sentences, paragraphs and articles. Thus, just as multiple choice tests of student knowledge are often supplemented by essay tests whose content is then judged by humans, we wished to evaluate the adequacy of LSA's representation of knowledge in complete passages of text. We could not have LSA write essays because it has no means for producing sentences. However, we were able to assess the accuracy with which LSA could extract and represent the knowledge expressed in essays written by students by simulating judgments about their content that were made by human readers (Landauer, Laham, Rehder, & Schreiner, in press).

In these tests, students were asked to write short essays to cover an assigned topic or to answer a posed question. In various experiments, the topics included anatomy and function of the heart, phenomena from introductory psychology, the history of the Panama Canal, and tolerance of diversity in America. In each case, LSA was first trained either on a large sample of instructional text from the same domain or, in the latter case, on combined text from the very large number of essays themselves, to produce a high-dimensional (100-350 dimensions in the various tests) semantic space. We then represented each essay simply as the vector average of the vectors for the words it contained. Two properties of these average vectors were then used to measure the quality and quantity of knowledge conveyed by an essay: (1) the similarity (measured as the cosine of the angle between vectors) of the student essay and one or more standard essays, and (2) the total amount of domain specific content, measured as the vector length.

In each case, two human experts independently rated the overall quality of each essay on a five or ten point scale. The judges were either university course instructors or professional exam readers from Educational Testing Service. The LSA measures were calibrated with respect to the judges' rating scale in several different ways, but because they gave nearly the same results only one will be described here. In this method, each student essay was compared to a large (90-200) set of essays previously scored by experts, and the ten most similar (by cosine) identified. The target essay was then assigned a "quality" score component consisting of the cosine-weighted average of the ten. A second, "relevant quantity", score component was the vector length of the student essay. Finally, regression on expert scores was used to weight the quality and quantity scores (However, the weights in all cases were so close to equal that merely adding them would have given comparable results). Calibration was performed on data independent of that used to evaluate the relation between LSA and expert ratings.

The correlation between the LSA score for an essay and that assigned by the average of the human readers was .80, .64, .XX and .84 for the four sets of exams. The comparable correlation between one reader and the other was .83, .65, .XX and .82, respectively. In the heart topic case, each student had also taken a carefully constructed "objective" test over the same material (a short answer test with near perfect scoring agreement). The correlation between the LSA essay score and the objective test was .81, the average correlation for the two expert readers .74.

A striking aspect of these results is that the LSA representations were based on analyses of the essays that took no account of word order, each essay was treated as a "bag of words". In extracting meaning from a text, human readers presumably rely on syntax as well as the mere combination of words it contains, yet they were no better at agreeing on an essay's quality or in assigning a score that predicted a performance on a separate test of knowledge. Apparently, either the relevant information conveyed by word order in sentences is redundant with the information that can be inferred from the combination of words in the essay, or the processes used by LSA and humans extract different but compensatingly useful information.

## 3.3 OTHER EVIDENCE

LSA has been compared with human knowledge in several additional ways, some confirming the correspondence, others indicating limitations. Here are some examples, all based on encyclopedia corpus training.

(1) Overall LSA similarity between antonyms (mean cos = .18) was equivalent to that between synonyms (mean cos = .17) in triplets sampled from an antonym/synonym dictionary (W & R Chambers, 1989), both of which significantly exceeded that for unrelated pairs (mean cos = .01; *ps* < .0001). However for antonym (but not for synonym) pairs a dominant dimension of difference could easily be extracted by computing a one dimensional unfolding using the LSA cosines from a set of words listed in Roget's (1992) thesaurus as related respectively to the two members of the pair.

(2) Anglin (1970) asked children and adults to sort words varying in concept relations and parts of speech. LSA word-word similarity correlates .50 with children and .32 with adults for the number of times they sorted two words together. Conceptual structure is reflected, but grammatical classification, strong in the adult data, is not.

(3) When people are asked to decide that a letter string is a word, they do so faster if they have just read a sentence that does not contain the word but implies a related concept (e.g. Till, Mross & Kintsch, 1988). LSA mirrors this result with high similarities between the same sentences and words. (Landauer & Dumais, 1997).

(4) People frequently make a logical error, called the conjunction error by Tversky and Kahneman (1974), in which they estimate that the probability that an object is a member of a class is greater than that it is a member of a superset class when the description of the object is "similar" to the description of the subset. For example, when told that "Linda is a young woman who is single, outspoken...deeply concerned with issues of discrimination and social justice," over 80% of even statistically sophisticated subjects rate it more likely that Linda is a feminist bank teller than that Linda is a bank teller (Tversky & Kahneman, 1980). LSA similarities between descriptions of people and occupations of this kind taken from Shafir, Smith and Osherson (1990) were computed as the cosine between the vector averages of words in the paired person-occupation descriptions. Conjunction error statements were more similar to the subset than superset statement in 12 out of 14 cases (*p*<.01), showing that LSA's representation of sentential meaning reflected similarity relations of the sort that have been hypothesized to underlie the conjunction fallacy in human judgment.

(5) A semantic subspace was constructed for words from natural kind and artifact categories whose differential preservation is characteristic of agnosias due to local damage from herpes simplex encephalitis (Warrington & Shallice, 1984). Principal components analysis of the similarities among these words as represented by LSA revealed that categories that tend to be lost contain words that are more highly inter-related than those in preserved categories (Laham, in press).

Of course, LSA does not capture all of the human knowledge conveyed by text. Some of the shortfall is probably due merely to the use of training corpora that are still imperfectly representative of the language experience of a typical person, and to lack of knowledge from non-textual sources. For example, in all these studies, less total text was used for LSA training than even a single educated adult would have read. However, a more fundamental restriction is that the analysis does not reflect order relations between words, and therefore cannot extract information that depends on syntax. Because the analysis discovers and represents only unsigned continuous similarities, it can be used to induce only certain classes of structural relations, not including ones that express Boolean, causal or other non-commutative logical relations. As we have seen, this lack does not prevent accurate simulation of human cognition in many cases, possibly because humans also

frequently rely on similarity rather than syntax-based, discrete logic (Tversky and Kahneman, 1983); however, it does limit the utility of the results for populating the symbolic data structures commonly used to represent knowledge in traditional AI. On the other hand, as examples in the next section show, continuous-valued similarity relations can be fruitfully applied if appropriate computational use is made of them.

## 4 SAMPLE APPLICATIONS

LSA has been used successfully in a variety of experimental applications, including the essay scoring techniques described earlier. Here are some additional examples:

(1) The technique has been used to improve automatic information retrieval by 20-30% over otherwise identical methods by allowing users' queries to match documents with the desired conceptual meaning but expressed in different words (Dumais, 1991, 1994).

(2) By training on corpora of translated text in which the words of corresponding paragraphs in the two languages are combined in the "bags of words", LSA has been able to provide at least as good retrieval when queries and documents are in a different language as when in the same language (Landauer and Littman, 1990).

(3) LSA-based measures of the similarity of student essays on a topic to instructional texts can predict how much an individual student will learn from a particular text (Wolfe *et al.*, in press; Rehder *et al.*, in press). To do this, the full set of student essays and the texts in question are aligned along a single dimension that best accommodates the LSA similarities among them. Estimates from one such experiment showed that using LSA to choose the optimal one of four texts for each student (a text that is slightly more sophisticated than the student) rather than assigning all students the overall best text (which LSA also picked correctly) increased the average amount learned by over 40%.

(4) LSA-based measures of conceptual similarity between successive sentences accurately predict differences in judged coherence and measured comprehensibility of text (Foltz, Kintsch and Landauer, in press).

## 5 SUMMARY

SVD-based learning of the structure underlying the use of words in meaningful contexts has been found capable of deriving and representing the similarity of meaning of words and text passages in a manner that accurately simulates corresponding similarity relations as reflected in several sorts of human judgments and behavior. The validity of the resulting representation of meaning similarities has been established in a variety of ways, and the utility of its knowledge representation illustrated by several educational and cognitive psychological research applications. It is obviously too early to assess whether the particular computational model is a true analog of the process used by the human brain to accomplish the same things. However, the basic process, the representation of myriad local associative relations between components and larger contexts of experience in a joint space of lower dimensionality, offers, for the first time, a candidate for such a mechanism that has been shown sufficient to approximate human knowledge acquisition from natural sources at natural scale.

### Acknowledgments

We thank members of the LSA research group at the University of Colorado for valuable collaboration and advice: Walter Kintsch, Bob Rehder, Mike Wolfe, & M. E. Shreiner. We especially acknowledge two participants from the Spring 1997 LSA seminar at CU whose unpublished work is described: Alan Sanfey (3.3.4) and Michael Emerson (3.3.1). Thanks also to Susan T. Dumais of Bellcore.

## References

Anglin, J. M. (1970). *The growth of word meaning*. Cambridge, MA: MIT.

Berry, M. W. (1992). Large scale singular value computations. *International Journal of Supercomputer Applications*, **6**, 13-49.

Deerwester, S., Dumais, S. T., Furnas, G. W., Landauer, T. K., & Harshman, R. (1990). Indexing By Latent Semantic Analysis. *Journal of the American Society For Information Science*, **41**, 391-407.

Dumais, S. T. (1991). Improving the retrieval of information from external sources. *Behavior Research Methods, Instruments and Computers*, **23**, 229-236.

Dumais, S. T. (1994). Latent semantic indexing (LSI) and TREC-2. In D. Harman (Ed.), *National Institute of Standards and Technology Text Retrieval Conference*. NIST special publication.

Foltz, P. W., Kintsch, W., & Landauer, T. K. (in press). Analysis of text coherence using Latent Semantic Analysis. *Discourse Processes*.

Laham, D. (in press). Latent Semantic Analysis approaches to categorization. *Proceedings of the Cognitive Science Society, 1997*.

Landauer, T. K., & Dumais, S. T. (1997). A solution to Plato's problem: The Latent Semantic Analysis theory of the acquisition, induction, and representation of knowledge. *Psychological Review*, **104**, 211-240.

Landauer, T. K., Foltz, P. W., & Laham, D. (1997). *Latent Semantic Analysis passes the test: knowledge representation and multiple-choice testing*. Manuscript in preparation.

Landauer, T. K., Laham, D., Rehder, B. & Schreiner, M .E. (in press). How well can passage meaning be derived without using word order: A comparison of Latent Semantic Analysis and humans. *Proceedings of the Cognitive Science Society, 1997*.

Landauer, T. K., & Littman, M. L. (1990). Fully automatic cross-language document retrieval using latent semantic indexing. In *Proceedings of the Sixth Annual Conference of the UW Centre for the New Oxford English Dictionary and Text Research* (pp. 31-38). Waterloo, Ontario: UW Centre for the New OED.

Myers, D. G. (1995). *Psychology, Fourth Edition*. NY, NY: Worth.

Rehder, B., Schreiner, M. E., Wolfe, B. W., Laham, D., Landauer, T. K., & Kintsch, W. (in press). Using Latent Semantic Analysis to assess knowledge: Some technical considerations. *Discourse Processes*.

Shafir, E., Smith, E. E., & Osherson, D. N. (1990). Typicality and reasoning judgments. *Memory & Cognition*, **3**, 229-239.

Till, R. E., Mross, E. F., & Kintsch. W. (1988). Time course of priming for associate and inference words in discourse context. *Memory and Cognition*, **16**, 283-299.

Tversky, A., & Kahneman, D. (1974). Judgment under uncertainty: Heuristics and biases. *Science*, **185**, 1124-1131.

Tversky, A., & Kahneman, D. (1980). Judgments of and by representativeness. In D. Kahneman, P. Slovic, & A. Tversky (Eds.), *Judgment under uncertainty: Heuristics and biases*. New York: Cambridge University Press.

Tversky, A., & Kahneman, D. (1983). Extensional versus intuitive reasoning: The conjunction fallacy in probability judgment. *Psychological Review*, **90**, 293-315.

Warrington, E. K., & Shallice, T. (1984). Category-specific semantic impairments. *Brain*, **107**, 829-853.

Wolfe, M. B., Schreiner, M. E., Rehder, B., Laham, D., Foltz, P. W., Kintsch, W., & Landauer, T. K. (in press). Learning from text: Matching readers and text by Latent Semantic Analysis. *Discourse Processes*.